# Optical Implementation of a Self-Organizing Feature Extractor

Dana Z. Anderson*, Claus Benkert, Verena Hebler, Ju-Seog Jang,
Don Montgomery, and Mark Saffman.

Joint Institute for Laboratory Astrophysics, University of Colorado and the
Department of Physics, University of Colorado, Boulder Colorado 80309-0440

## Abstract

We demonstrate a self-organizing system based on photorefractive ring oscillators. We employ the system in two ways that can both be thought of as feature extractors; one acts on a set of images exposed repeatedly to the system strictly as a linear feature extractor, and the other serves as a signal demultiplexer for fiber optic communications. Both systems implement unsupervised competitive learning embedded within the mode interaction dynamics between the modes of a set of ring oscillators. After a training period, the modes of the rings become associated with the different image features or carrier frequencies within the incoming data stream.

## 1 Introduction

Self-organizing networks (Kohonen, Hertz, Domany) discover features or qualities about their input environment on their own; they learn without a teacher making explicit what is to be learned. This property reminds us of several ubiquitous behaviors we see in the physical and natural sciences such as pattern formation, morphogenesis and phase transitions (Domany). While in the natural case one is usually satisfied simply to *analyze* and understand the behavior of a self-organizing system, we usually have a specific function in mind that we wish a neural network to perform. That is, in the network case we wish to *synthesize* a system that will perform the desired function. Self-organizing principles are particularly valuable when one does not know ahead of time exactly what to expect from the input to be processed and when it is some property of the input itself that is of interest. For example, one may wish to determine some quality about the input statistics - this one can often do by applying self-organization principles. However, when one wishes to attribute some meaning to the data, self-organization principles are probably poor candidates for this task.

It is the behavioral similarity between self-organizing network models and physical systems that has lead us to investigate the possibility of implementing a self-organizing network function by designing the dynamics for a set of optical oscillators. Modes of sets of oscillators undergo competition (Anderson, Benkert) much like that employed in competitive learning network models. Using photorefractive elements, we have tailored the dynamics of the mode interaction to perform a learning task. A physical optical implementation of self-organized learning serves two functions. Unlike a computer simulation, the physical system *must* obey certain physical laws just like a biological system does. We have in mind the consequences of energy conservation, finite gain and the effects of noise. Therefore, we might expect to learn something about general principles applicable to biological systems from our physical versions. Second, there are some applications where an optical system serves as an ideal "front end" to signal processing.

Here we take a commonly used supervised approach for extracting features from a stream of images and demonstrate how this task can be done in a self-organizing manner. The conventional approach employs a holographic correlator (Vander Lugt). In this technique, various patterns are chosen for recognition by the optical system and then recorded in holographic media using angle-encoded reference beams. When a specific pattern is presented to the holographic correlator, the output is determined by the correlation between the presented pattern and the patterns that have been recorded as holograms during the 'learning phase'. The angles and intensities of the reconstructed reference beams identify the features present in the pattern. Because the processing time-scale in holographic systems is determined by the time necessary for light to scatter off of the holographic grating, the optical correlation takes place virtually instantaneously. It is the speed of this correlation that makes the holographic approach so interesting.

While its speed is an asset, the holographic correlator approach to feature extraction from images is a supervised approach to the problem: an external supervisor must choose the relevant image features to store in the correlator holograms. Moreover the supervisor must provide an angle-encoded reference beam for each stored feature. For many applications, it is desirable to have an adaptive system that has the innate capacity to discover, in an unsupervised fashion, the underlying structure within the input data.

A photorefractive ring resonator circuit that learns to extract spatially orthogonal features from images is illustrated schematically in figure 1. The resonator rings in figure 1 are constructed physically from optical fibers cables. The resonator is self-starting and is pumped by images containing the input data (White). The resonator learns to associate each feature in the input data set with one and only one of the available resonator rings. In other words, when the proper feature is present in the input data, the resonator ring with which it has become associated will light up. When this feature is absent from the input data, the corresponding resonator ring will be dark.

The self-organizing capabilities of this system arise from the nonlinear dynam-

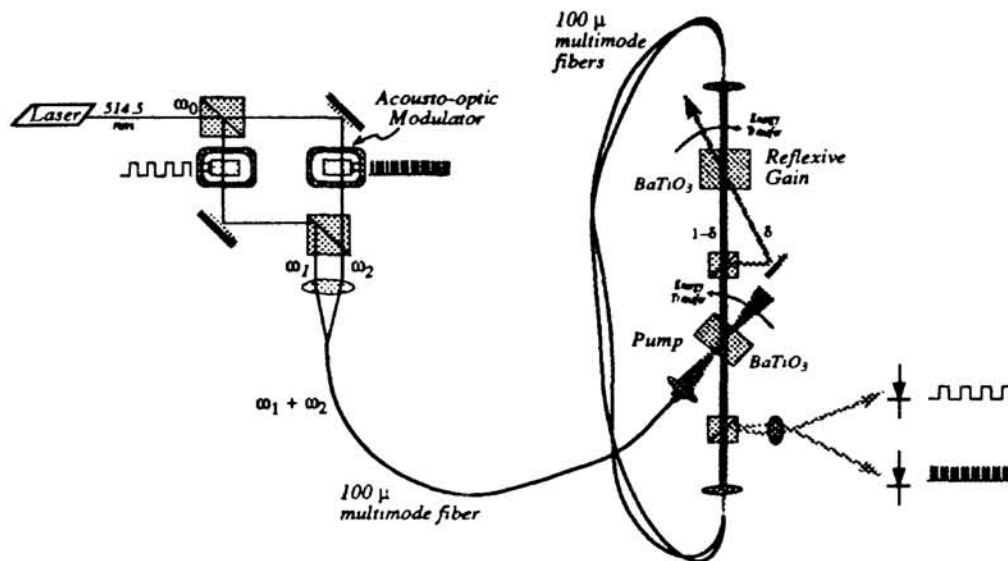

Figure 1: Schematic diagram of the self-organizing photorefractive ring resonator. The two signal frequencies, $\omega_1$ amd $\omega_2$, are identical when the circuit is used as a feature extractor and are separated by 280 MHz when the system is used as a frequency demultiplexer.

ics of competition between resonator modes for optical energy within the common photorefractive pump crystal (Benkert). We have used this system to accomplish two optical signal processing tasks. In the first case, the resonator can learn to distinguish between two spatially orthogonal input images that are impressed on the common pump beam in a piece-wise constant fashion. In the second case, frequency demultiplexing of a composite input image constructed from two spatially orthogonal image components of different optical frequencies can be accomplished (Saffman, 1991b). In both cases, the optical system has no *a priori* knowledge of the input data and self-discovers the important structural elements.

## 2 A Self-Organizing Photorefractive Ring Resonator

The experimental design that realizes an optical self-organizing feature extractor is shown in figure 1. The optical system consists of a two ring, multimode, unidirectional photorefractive ring resonator in which the rings are spatially distinct. The resonator rings are defined by loops of 100 μ core multimode optical fiber. The gain for both modes is provided by a common $BaTiO_3$ crystal that is pumped by optical images presented as speckle patterns from a single 100 μ multimode optical fiber. The light source is a single frequency argon-ion laser operating at 514.5 nm. The second $BaTiO_3$ crystal provides reflexive coupling within the resonator, which ensures that each resonator ring becomes associated with only one input feature.

The input images are generated by splitting the source beam and passing it through two acousto-optic modulator cells. The optical signals generated by the acousto-optic modulators are then focused into a single 1.5 meter long step-index, 100 μ core, multimode optical fiber. The difference in the angle of incidence for the two signal beams at the fiber end face is sufficient to ensure that the corresponding speckle pattern images are spatially orthogonal (Saffman,

1991a). The acousto-optic cells are used in a conventional fashion to shift the optical frequency of the carrier signal, and are also used as shutters to impress time modulated information on the input signals. When the resonator is operating as a feature extractor, both input signals are carried on the same optical frequency, but are presented to the resonator sequentially. The presentation cycle time of 500 Hz was chosen to be much smaller than the characteristic time constant of the $BaTiO_3$ pump crystal. When operating as a frequency demultiplexer, the acousto-optic modulators shift the optical carrier frequencies of the input signals such that they are separated by 280 MHz. The two input carrier signals are time modulated and mixed into the optical fiber to form a composite image composed of two spatially orthogonal speckle patterns having different optical frequencies. This composite image is used as the pump beam for the resonator.

## 3 Unsupervised Competitive Learning

Correlations between the optical electric fields in images establish the criterion for a measure of similarity between different image features. The best measure of these correlations is the inner product between the complex-valued spatial electric field distribution across the input images,

$$S_{12} = \left| \iint d^2x E_1^*(x) E_2(x) \right|$$

When $S_{12} = 0$ the images are uncorrelated and we define such images as spatially orthogonal. When the resonator begins to oscillate, neither resonator ring has any preference for a particular input feature or frequency. The system modes have no internal bias (i.e., no a priori knowledge) for the input data. As the gain for photorefractive two-beam coupling in the common $BaTiO_3$ pump crystal saturates, the two resonator rings begin to compete with each other for the available pump energy. This competitive coupling leads to 'winner-takes-all' dynamics in the resonator in which each resonator ring becomes associated with one or the other spatially orthogonal input images. In other words, the rings become labels for each spatially orthogonal feature present in the input image set.

Phenomenologically, the dynamics of this mode competition can be described by Lotka-Volterra equations (Benkert, Lotka, Volterra),

$$\frac{dI_{i,p}}{dt} = I_{i,p}\left(\alpha_{i,p} - \beta_{i,p}I_{i,p} - \sum_{j,l} \theta_{i,p;j,l}I_{j,l}\right)$$

Where $I_{i,p}$ is the intensity of the oscillating energy in ring $i$ due to energy transferred from the input feature $p$, $\alpha_{i,p}$ is the gain for two-beam coupling between ring $i$ and feature $p$, $\beta_{i,p}$ is the self-saturation coefficient, and $\theta_{i,p;j,l}$ are the cross-saturation coefficients. The self-organizing dynamics are determined by the values of the cross coupling coefficients. Thus the competitive learning algorithm that drives the self-organization in this optical system is embedded

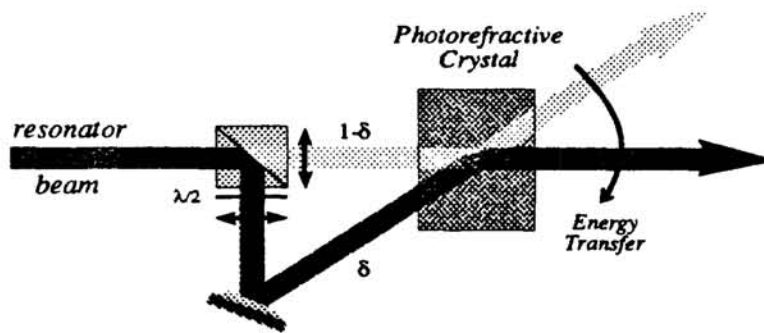

Figure 2: Reflexive gain interaction. A fraction, $\delta$, of the incident intensity is removed from the resonator beam, and then coupled back into itself by photorefractive two beam coupling. This ensures 'Winner-takes-all' competitive dynamics between the resonator rings.

within the nonlinear dynamics of mode competition in the pump crystal.

Once the system has learned, the spatially orthogonal features in the training set are represented as holograms in the $BaTiO_3$ pump crystal. These holograms act as linear projection operators, and any new image constructed from features in the training set will be projected in a linear fashion onto the learned feature basis set. The relative intensity of light oscillating in each ring corresponds to the fraction of each learned feature in the new image. Thus, the resonator functions as a feature extractor (Kohonen).

## 4 Reflexive Gain

If each resonator ring was single mode, then competitive dynamics in the common pump crystal would be sufficient for feature extraction. However, a multimode ring system allows stability for certain pathological feature extracting states. The multimode character of each resonator ring can permit simultaneous oscillation of two spatially orthogonal modes within a single ring. Ostensibly, the system is performing feature extraction, but this form of output is not useful for further processing. These pathological states are excluded by introducing reflexive gain into the cavity.

Any system that twists back upon itself and closes a loop is referred to as *reflexive* (Hofstadter, pg. 3). A reflexive gain interaction is achieved by removing a portion of the oscillating energy from each ring and then coupling it back into the same ring by photorefractive two-beam coupling, as illustrated in figure 2. The standard equations for photorefractive two-beam coupling (Kukhtarev, Hall) can be used to derive an expression for the steady-state transmission, T, through the reflexive gain element in terms of the number of spatially orthogonal modes, N, that are oscillating simultaneously within a single ring,

$$T = \left[1 + \frac{\delta}{1-\delta}\exp\left(\frac{-G_0}{N}\right)\right]^{-1}$$

Here, $\exp(G_0)$ is the small signal gain and $\delta$ is the fraction of light removed

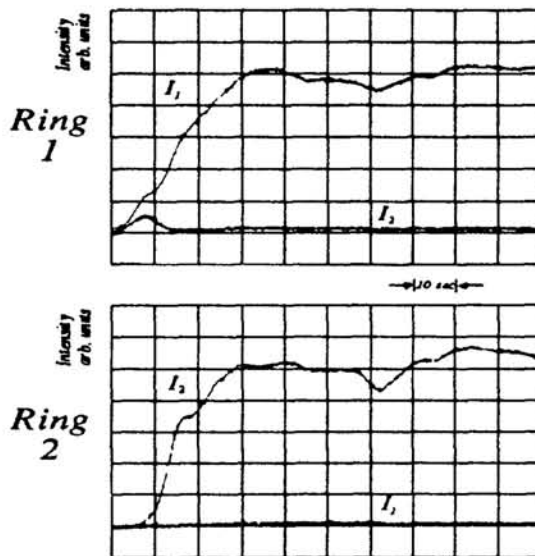

Figure 3: Time evolution of the intensities within each resonator ring due to $\omega_1$ ($I_1$) and $\omega_2$($I_2$). After about 30 seconds, the system has learned to demultiplex the two input frequencies. Ring 1 has become associated with $\omega_1$ and Ring 2 has become associated with $\omega_2$. The contrast ratio between $I_1$ and $I_2$ in each ring is about 40:1.

from the resonator. The transmission decreases for $N > 1$ causing larger cavity losses for the case of simultaneous oscillation of spatially orthogonal modes within a single ring. Therefore, the system favors 'winner-takes-all' dynamics over other pathological feature extracting states.

## 5 Experimental Results

The self-organizing dynamics within the optical circuit require several seconds to reach steady state. In the case of frequency demultiplexing, the dynamical evolution of the system was observed by detecting the envelopes of the carrier modulation, as shown in figure 3. In the case of the feature extractor, transient system dynamics were observed by synchronizing the observation with the modulation of one feature or the other, as shown in figure 4. The frequency demultiplexing (figure 3) and feature extracting (figure 4) states develop a high contrast ratio and are stable for as long as the pump beam is present. Measurements with a spectrum analyzer show an output contrast ratio of better than 40:1 in the frequency demultiplexing case.

The circuit described here extracts spatially orthogonal features while contin-

Figure 4: Time evolution of the intensities in each resonator ring due to the two input pictures. The system requires about 30 seconds to learn to extract features from the input images. Picture 1 is associated with Ring 1 and picture 2 is associated with Ring2.

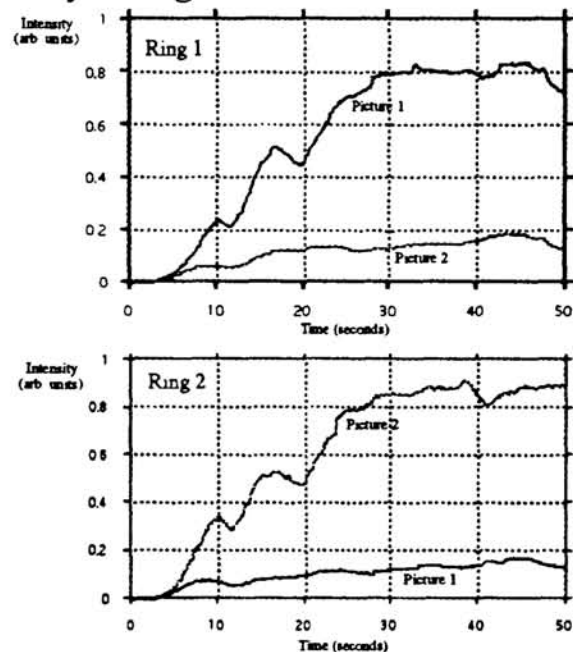

uously adapting to slow variations in the spatial mode superposition due to drifts in the carrier frequency or perturbations to the fibers. Thus, the system is adaptive as well as unsupervised.

# 6 Summary

An optical implementation of a self-organizing feature extractor that is adaptive has been demonstrated. The circuit exhibits the desirable dynamical property that is often referred to in the parlance of the neural networks as 'unsupervised learning'. The essential properties of this system arise from the nonlinear dynamics of mode competition within the optical ring resonator. The learning algorithm is embedded in these dynamics and they contribute to its capacity to adapt to slow changes in the input signal. The circuit learns to associate different spatially orthogonal images with different rings in an optical resonator. The learned feature set can represent orthogonal basis vectors in an image or different frequencies in a multiplexed optical signal. Because a wide variety of information can be encoded onto the input images presented to the feature extractor described here, it has the potential to find general application for tasks where the speed and adaptability of self-organizing and all-optical processing is desirable.

# Acknowledgements

We are grateful for the support of both the Office of Naval Research, contract #N00014-91-J-1212 and the Air Force Office of Scientific Research, contract #AFOSR-90-0198. Mark Saffman would like to acknowledge support provided by a U.S. Air Force Office of Scientific Research laboratory graduate fellowship.

# References

D.Z. Anderson and R. Saxena, *Theory of Multimode Operation of a Unidirectional Ring Oscillator having Photorefractive Gain: Weak Field Limit*, J. Opt. Soc. Am. B, **4**, 164 (1987).

C. Benkert and D.Z. Anderson, *Controlled competitive dynamics in a photorefractive ring oscillator: "Winner-takes-all" and the "voting-paradox" dynamics*, Phys. Rev. A, **44**, 4633 (1991).

E. Domany, J.L. van Hemmen and K. Schulten, eds., *Models of Neural Networks*; Springer-Verlag (1991).

T.J. Hall, R. Jaura, L.M. Connors and P.D. Foote, *The Photorefractive Effect - A Review*; Prog. Quant. Electr., **10**, 77 (1985).

J. Hertz, A. Krogh and R.G. Palmer, *Introduction to the Theory of Neural Computation*; Addison-Wesley (1991).

D. R. Hofstadter, *Metamagical Themas: Questing for the Essence of Mind and Pattern*; Bantam Books (1985).

T. Kohonen, *Self-Organization and Associative Memory*, 3rd Edition; Springer-Verlag (1989).

N. V. Kukhtarev, V.B. Markov, S.G. Odulov, M.S. Soskin and V.L. Vinetskii, *Holographic Storage in Electrooptic Crystals. I. Steady State*; Ferroelectrics, **22**, 949 (1979).

N. V. Kukhtarev, V.B. Markov, S.G. Odulov, M.S. Soskin and V.L. Vinetskii, *Holographic Storage in Electrooptic Crystals. II. Beam Coupling - Light Amplification;* Ferroelectrics, **22**, 961 (1979).

A.J. Lotka, *Elements of Physical Biology*; Baltimore (1925).

M. Saffman and D.Z. Anderson, *Mode multiplexing and holographic demultiplexing communications channels on a multimode fiber*, Opt. Lett., **16**, 300 (1991a).

M. Saffman, C. Benkert and D.Z. Anderson, *Self-organizing photorefractive frequency demultiplexer*, Opt. Lett., **16**, 1993 (1991b).

A. Vander Lugt, *Signal Detection by Complex Spatial Filtering*; IEEE Trans. Inform. Theor., **IT-10**, 139 (1964).

V. Volterra, *Leçons sur la Théorie Mathematiques de la Lutte pour la Vie*; Gauthier-Villars (1931).

J.O. White, M. Cronin-Golomb, B. Fischer, and A. Yariv, *Coherent Oscillation by Self-Induced Gratings in the Photorefractive Crystal BaTiO₃*; Appl. Phys. Lett., **40**, 450 (1982).
